# Burst Synchronization Without Frequency-Locking in a Completely Solvable Network Model

**Heinz Schuster**
Institut für theoretische Physik
Universität Kiel
Olshausenstraße 40
2300 Kiel 1, Germany

**Christof Koch**
Computation and Neural System Program
California Institute of Technology
Pasadena, California 91125, USA

## Abstract

The dynamic behavior of a network model consisting of all-to-all excitatory coupled binary neurons with global inhibition is studied analytically and numerically. We prove that for random input signals, the output of the network consists of synchronized bursts with apparently random intermissions of noisy activity. Our results suggest that synchronous bursts can be generated by a simple neuronal architecture which amplifies incoming coincident signals. This synchronization process is accompanied by dampened oscillations which, by themselves, however, do not play any constructive role in this and can therefore be considered to be an epiphenomenon.

## 1 INTRODUCTION

Recently synchronization phenomena in neural networks have attracted considerable attention. Gray *et al.* (1989, 1990) as well as Eckhorn *et al.* (1988) provided electrophysiological evidence that neurons in the visual cortex of cats discharge in a semi-synchronous, oscillatory manner in the 40 $Hz$ range and that the firing activity of neurons up to 10 $mm$ away is phase-locked with a mean phase-shift of less than 3 $msec$. It has been proposed that this phase synchronization can solve the binding problem for figure-ground segregation (von der Malsburg and Schneider, 1986) and underly visual attention and awareness (Crick and Koch, 1990).

A number of theoretical explanations based on coupled (relaxation) oscillator models have been proposed for burst synchronization (Sompolinsky *et al.*, 1990). The crucial issue of phase synchronization has also recently been addressed by Bush and Douglas (1991), who simulated the dynamics of a network consisting of bursty, layer V pyramidal cells coupled to a common pool of basket cells inhibiting all pyramidal cells.[1] Bush and Douglas found that excitatory interactions between the pyramidal cells increases the total neural activity as expected and that global inhibition leads to synchronized bursts with random intermissions. These population bursts appear to occur in a random manner in their model. The basic mechanism for the observed burst synchronization is hidden in the numerous anatomical and biophysical details of their model. These, and the related observation that to date no strong oscillations have been recorded in the neuronal activity in visual cortex of awake monkeys, prompted us to investigate how phase synchronization can occur in the absence of frequency locking.

## 2   A COINCIDENCE NETWORK

We consider $n$ excitatory coupled binary McCulloch-Pitts (1943) neurons whose output $x_i^{t+1} \in [0,1]$ at time $t+1$ is given by:

$$x_i^{t+1} = \sigma\left[\frac{\omega}{n}\sum_i x_i^t + \xi_i^t - \theta\right] \tag{1}$$

Here $\omega/n > 1$ is the normalized excitatory all-to-all synaptic coupling, $\xi_i^t$ represents the external binary input and $\sigma[z]$ is the Heaviside step function, such that $\sigma[z] = 1$ for $z > 0$ and 0 elsewhere. Each neuron has the same dynamic threshold $\theta > 0$. Next we introduce the fraction $m^t$ of neurons which fire simultaneously at time $t$:

$$m^t = \frac{1}{n}\sum_i x_i^t \tag{2}$$

In general, $0 \leq m^t \leq 1$; only if every neuron is active at time $t$ do we have $m^t = 1$. By summing eq. (1) we obtain the following equation of motion for our simple network.

$$m^{t+1} = \frac{1}{n}\sum_i \sigma[\omega m^t + \xi_i^t - \theta] \tag{3}$$

The behavior of this (n+1)-state automata is fully described by the phase-state diagram of Figure 1. If $\theta > 1$ and $\theta/\omega > 1$, the output of the network $m^t$ will vary with the input until at some time $t'$, $m^{t'} = 0$. Since the threshold $\theta$ is always larger than the input, the network will remain in this state for all subsequent times. If $\theta < 1$ and $\theta/\omega < 1$, the network will drift until it comes to the state $m^{t'} = 1$. Since subsequent $\omega m^t$ is at all times larger than the threshold, the network remains latched at $m^t = 1$. If $\theta > 1$, but $\theta/\omega < 1$, the network can latch in either the $m^t = 0$ or the $m^t = 1$ state and will remain there indefinitely. Lastly, if $\theta < 1$, but $\theta/\omega > 1$, the threshold is by itself not large enough to keep the network latched

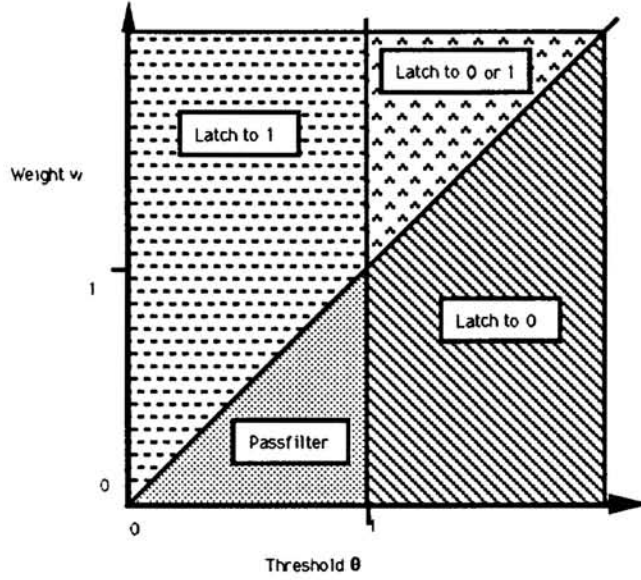

Figure 1: Phase diagram for the network described by eq. (3). Different regions correspond to different stationary output states $m^t$ in the long time limit.

into the $m^t = 1$ state. Defining the normalized input activity as

$$s^t = \frac{1}{n}\sum_i \xi_i^t \tag{4}$$

with $0 \leq s^t \leq 1$, we see that in this part of phase space $m^{t+1} = s^t$, and the output activity faithfully reflects the input activity at the previous time step.

Let us introduce an adaptive time-dependent threshold, $\theta^t$. We assume that $\theta^t$ remains at its value $\theta < 1$ as long as the total activity remains less than 1. If, however, $m^t = 1$, we increase $\theta^t$ to a value larger than $\omega + 1$. This has the effect of resetting the activity of the entire network to 0 in the next time step, i.e., $m^{t+1} = (1/n)\sum_i \sigma(\omega + \xi_i - (\omega + 1 + \epsilon)) = 0$. The threshold will then automatically reset itself to its old value:

$$m^{t+1} = \frac{1}{n}\sum_i \sigma[\omega m^t + \xi_i^t - \theta(m^t)] \tag{5}$$

with

$$\theta(m^t) = \begin{cases} \theta < 1 & \text{for } m^t < 1 \\ > \omega + 1 & \text{for } m^t = 1 \end{cases}$$

Therefore, we are operating in the topmost left part of Fig. 1 but preventing the network from latching to $m^t = 1$ by resetting it. Such a dynamic threshold bears some similarities to the models of Horn and Usher (1990) and others, but is much simpler. Note that $\theta(m^t)$ exactly mimics the effect of a common inhibitory neuron which is only excited if all neurons fire simultaneously.

Our network now acts as a coincidence detector, such that all neurons will "fire" at time $t + 2$, i.e., $m_1^{t+2} = 1$ if at least $k$ neurons receive at time $t$ a "1" as input. $k$ is the smallest integer with $k > \theta \cdot n/\omega$. The threshold $\theta(m^t)$ is then transiently increased and the network is reset and the game begins anew. In other words, the

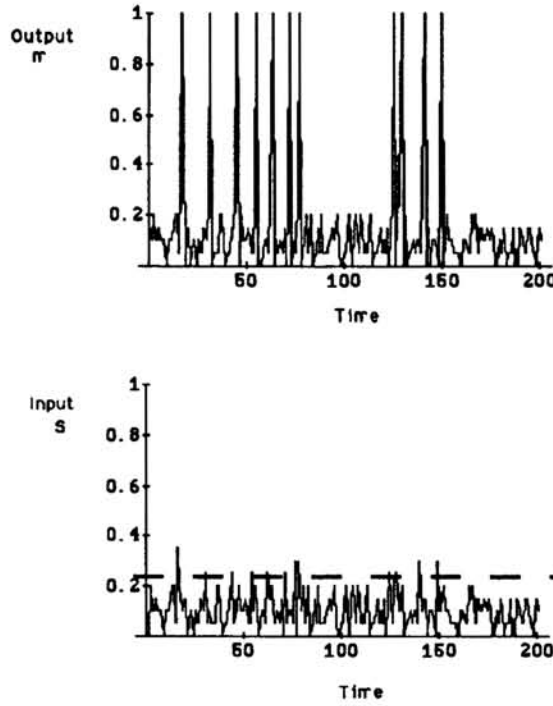

Figure 2: Time dependence of the fraction $m^t$ of output neurons which fire simultaneously compared to the corresponding fraction of input signals $s^t$ for $n = 20$ and $\theta/\omega = 0.225$. The input variables $\xi_i^t$ are independently distributed according to $P(\xi_i^t) = p\delta(\xi_i^t - 1) + (1 - p)\delta(\xi_i^t)$ with $p = 0.1$. If more than five input signals with $\xi_i^t = 1$ coincide, the entire population will fire in synchrony two time steps later, i.e. $m^{t+2} = 1$. Note the "random" appearance of the interburst intervals.

network detects coincidences and signals this by a synchronized burst of neuronal activity followed by a brief respite of activity (Figure 2).

The time dependence of $m^t$ given by eq. (5) can be written as:

$$m^{t+1} = \begin{cases} s^t & \text{for } 0 \le m^t < \frac{\theta}{\omega} \\ 1 & \text{for } \frac{\theta}{\omega} \le m^t < 1 \\ 0 & \text{for } m^t = 1 \end{cases} \qquad (6)$$

By introducing functions $A(m)$, $B(m)$, $C(m)$ which take on the value 1 in the intervals specified for $m = m^t$ in eq. (6), respectively, and zero elsewhere, $m^{t+1}$ can be written as:

$$m^{t+1} = s^t A(m^t) + 1 \cdot B(m^t) + 0 \cdot C(m^t) \qquad (7)$$

This equation can be iterated, yielding an explicit expression for $m^t$ as a function of the external inputs $s^{t+1}, \ldots s^0$ and the initial value $m^0$:

$$m^t = (s^{t-1}, 1, 0)M(s^{t-2})\ldots M(s^0)\begin{pmatrix} A(m^0) \\ B(m^0) \\ C(m^0) \end{pmatrix} \qquad (8)$$

with the matrix

$$M(s) = \begin{pmatrix} A(s) & 0 & 1 \\ B(s) & 0 & 0 \\ C(s) & 1 & 0 \end{pmatrix}$$

Eq. (8) shows that the dynamics of the network can be solved explicitly, by iteratively applying $M$, $t - 1$ number of times to the initial network configuration.

## 3   DISTRIBUTION OF BURSTS AND TIME CORRELATIONS

The synchronous activity at time $t$ depends on the specific realization of the input signals at different times (eq. 8). In order to get rid of this ambiguity we resort to averaged quantities where averages are understood over the distribution $\hat{P}\{s^t\}$ of inputs $s^t = \frac{1}{n}\sum_{i=1}^{n}\xi_i^t$. A very useful averaged quantity is the probability $P^t(m)$, describing the fraction $m$ of simultaneously firing neurons at time $t$. $P^t(m)$ is related to the probability distribution $P\{s^t\}$ via:

$$P^t(m) = \langle\delta[m - m^t\{s^{t-1},\dots s^0\}]\rangle \tag{9}$$

where $\langle\dots\rangle$ denotes the average with respect to $\hat{P}\{s^t\}$ and $m^t\{s^{t-1},\dots s^0\}$ is given by eq. (8). If the input signals $\xi_i^t$ are uncorrelated in time, $m^{t+1}$ depends according to eq. (7) only on $m^t$, and the time evolution of $P^t(m)$ can be described by the Chapman-Kolmogorov equation. We then find:

$$P^t(m) = P^\infty(m) + [P^0(m) - P^\infty(m)] \cdot f(t) \tag{10}$$

where

$$P^\infty(m) = \frac{1}{1 + 2\eta}[\hat{P}(m) + \eta\delta(m - 1) + \eta\delta(m)] \tag{11}$$

is the limiting distribution which evolves from the initial distribution $P^0(m)$ for large times, because the factor $f(t) = \eta^{\frac{t}{2}}\cos(\Omega t)$, where $\Omega = \pi - \arctan[\sqrt{4\eta - \eta^2}/\eta]$, decays exponentially with time and $\eta = \int_0^1 \hat{P}(s)B(s)ds = \int_{\theta/\omega}^1 \hat{P}(s)ds$. Notice that $0 \le \eta \le 1$ holds (for more details, see Koch and Schuster, 1992).

The limiting equilibrium distribution $P^\infty(m)$ evolves from the initial distribution $P^0(m)$ in an oscillatory fashion, with the building up of two delta-functions at $m = 1$ and $m = 0$ at the expense of $P^0(m)$. This signals the emergence of synchronous bursts, i.e., $m^t = 1$, which are always followed at the next time-step by zero activity, i.e., $m^{t+1} = 0$ (see also Fig. 2). The equilibrium value for the mean fraction of synchronized neurons is

$$\langle m^\infty\rangle = \int_0^1 dm P^\infty(m)m = \frac{\langle s\rangle + \eta}{1 + 2\eta} \tag{12}$$

which is larger than the initial value $\langle s\rangle = \int_0^1 ds P(s)s$, for $\langle s\rangle < \frac{1}{2}$, indicating an increase in synchronized bursting activity.

It is interesting to ask what type of time correlations will develop in the output of our network if it is stimulated with uncorrelated noise, $\xi_i^t$. The autocovariance function is

$$C(\tau) = \lim_{t\to\infty}[\langle m^{t+\tau}m^t\rangle - \langle m^t\rangle^2], \tag{13}$$

can be computed directly since $m^t$ and $P^\infty(m)$ are known explicitly. We find

$$C(\tau) = \delta_{\tau,0}C_0 + (1 - \delta_{\tau,0})C_1\eta^{|\tau|/2}\cos(\Omega\tau + \varphi) \tag{14}$$

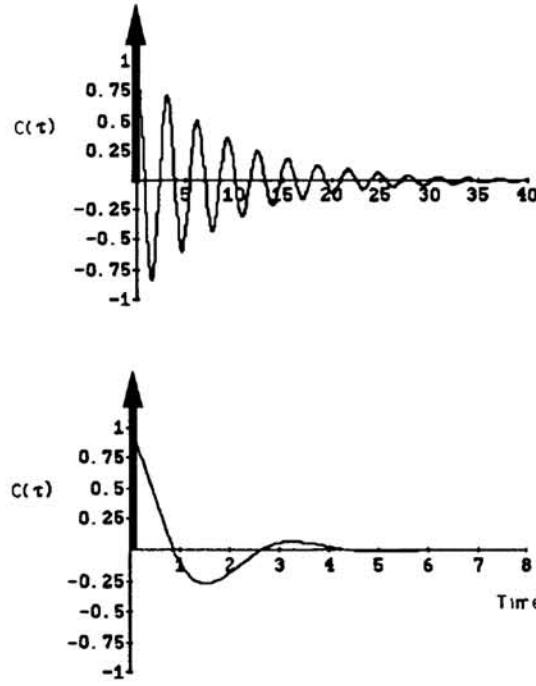

Figure 3: Time dependence of the auto-covariance function $C(\tau)$ for two different values of $\eta = \int_{\theta/\omega}^1 ds \hat{P}(s)$. The top figure corresponds to $\eta = 0.8$ and a period $T = 3.09$, while the bottom correlation function is for $\eta = 0.2$ with an associated $T = 3.50$. Note the different time-scales.

with $\delta_{\tau,0}$ the Kroneker symbol ($\delta_{\tau,0} = 1$ for $\tau = 0$ and 0 else). Figure 3 shows that $C(\tau)$ consists of two parts. A delta peak at $\tau = 0$ which reflects random uncorrelated bursting and an oscillatory decaying part which indicates correlations in the output. The period of the oscillations, $T = 2\pi/\Omega$, varies monotonically between $3 \le T \le 4$ as $\theta/\omega$ moves from zero to one. Since $\eta$ is given by $\int_{\theta/\omega}^1 \hat{P}(s)ds$, we see that the strengths of these oscillations increases as the excitatory coupling $\omega$ increases. The emergence of periodic correlations can be understood in the limit $\theta/\omega \to 0$, where the period $T$ becomes three (and $\eta = \int_0^1 \hat{P}(s)ds = 1$), because according to eq. (6), $m^t = 0$ is followed by $m^{t+1} = s^t$ which leads for $\theta/\omega \to 0$ always to $m^{t+2} = 1$ followed by $m^{t+3} = 0$. In other words, the temporal dynamics of $m^t$ has the form $0s^1 10s^4 10s^7 10s^{10} 10....$ In the opposite case of $\theta/\omega \to 1$, $\eta$ converges to 0 and the autocovariance function $C(\tau)$ essentially only contains the peak at $\tau = 0$. Thus, the output of the network ranges from completely uncorrelated noise for $\theta/\omega \approx 1$ to correlated periodic bursts for $\frac{\theta}{\omega} \to 0$. The power spectrum of the system is a broad Lorentzian centered at the oscillation frequency, superimposed onto a constant background corresponding to uncorrelated neural activity.

It is important to discuss in this context the effect of the size $n$ of the network. If the input variables $\xi_i^t$ are distributed independently in time and space with probabilities $\tilde{P}_i(\xi_i^t)$, then the distribution $\hat{P}(s)$ has a width which decreases as $1/\sqrt{n}$ as $n \to \infty$. Therefore, in a large system $\eta = \int_{\theta/\omega}^1 \hat{P}(s)ds$ is either 0 if $\theta/\omega > \langle s \rangle$ or 1 if $\theta/\omega < \langle s \rangle$, where $\langle s \rangle$ is the mean value of $s$, which coincides for $n \to \infty$ with the maximum of

$\hat{P}(s)$. If $\eta = 0$ the correlation function is a constant according to eq. (14), while the system will exhibit undamped oscillations with period 3 for $\eta = 1$. Therefore, the irregularity of the burst intervals, as shown, for instance, in Fig. 2, is for independent $\xi_i^t$ a finite size effect. Such synchronized dephasing due to finite size has been reported by Sompolinsky *et al.* (1989).

However, for biologically realistic correlated inputs $\xi_i^t$, the width of $\hat{P}(s)$ can remain finite for $n \gg 1$. For example, if the inputs $\xi_1^t, \ldots, \xi_n^t$ can be grouped into $q$ correlated sets $\xi_1^t \ldots \xi_1^t, \xi_2^t \ldots \xi_2^t, \ldots, \xi_q^t \ldots \xi_q^t$, with finite $q$, then the width of $\hat{P}(s)$ scales like $1/\sqrt{q}$. Our model, which now effectively corresponds to a situation with a finite number $q$ of inputs, leads in this case to irregular bursts which mirror and amplify the correlations present in the input signals, with an oscillatory component superimposed due to the dynamical threshold.

## 4   CONCLUSIONS AND DISCUSSION

We here suggest a mechanism for burst synchronization which is based on the fact that excitatory coupled neurons fire in synchrony whenever a sufficient number of input signals coincide. In our model, common inhibition shuts down the activity after each burst, making the whole process repeatable, without entraining any signals. It is rather satisfactory to us that our simple model shows qualitative similar dynamic behavior of the much more detailed biophysical simulations of Bush and Douglas (1991). In both models, all-to-all excitatory coupling leads—together with common inhibition—to burst synchronization without frequency locking. In our analysis we updated all neurons in parallel. The same model has been investigated numerically for serial (asynchronous) updating, leading to qualitatively similar results.

The output of our network develops oscillatory correlations whose range and amplitude increases as the excitatory coupling is strengthened. However, these oscillations do not depend on the presence of any neuronal oscillators, as in our earlier models (e.g., Schuster and Wagner, 1990; Niebur *et al.*,1991). The period of the oscillations reflects essentially the delay between the inhibitory response and the excitatory stimulus and only varies little with the amplitude of the excitatory coupling and the threshold. The crucial role of inhibitory interneurons in controlling the $40\,Hz$ neuronal oscillations has been emphasized by Wilson and Bower (1992) in their simulations of olfactory and visual cortex. Our model shows complete synchronization, in the sense that all neurons fire at the same time. This suggests that the occurrence of tightly synchronized firing activity across neurons is more important for feature linking and binding than the locking of oscillatory frequencies. Since the specific statistics of the input noise is, via coincidence detection, mirrored in the burst statistics, we speculate that our network—acting as an amplifier for the input noise—can play an important role in any mechanism for feature linking that exploits common noise correlations of different input signals.

### Acknowledgements

We thank R. Douglas for stimulating discussions and for inspiring us to think about this problem. Our collaboration was supported by the Stiftung Volkswagenwerk. The research of C.K. is supported by the National Science Foundation, the James

McDonnell Foundation, and the Air Force Office of Scientific Research.

## Footnotes

[1]This model bears similarities to Wilson and Bower's (1992) model describing the origin of phase-locking in olfactory cortex.

## References

Bush, P.C. and Douglas, R.J. "Synchronization of bursting action potential discharge in a model network of neocortical neurons." *Neural Computation* **3**: 19–30, 1991.

Crick, F. and Koch, C. "Towards a neurobiological theory of consciousness." *Seminars Neurosci.* **2**: 263-275, 1990.

Eckhorn, R., Bauer, R., Jordan, W., Brosch, M., Kruse, W., Munk, M. and Reitboeck, H.J. "Coherent oscillations: a mechanism of feature linking in the visual cortex?" *Biol. Cybern.* **60**: 121–130, 1988.

Gray, C.M., Engel, A.K., König, P. and Singer, W. "Stimulus-dependent neuronal oscillations in cat visual cortex: Receptive field properties and feature dependence." *Eur. J. Neurosci.* **2**: 607–619, 1990.

Gray, C.M., Konig, P., Engel, A.K. and Singer, W. "Oscillatory response in cat visual cortex exhibits inter-columnar synchronization which reflects global stimulus attributes." *Nature* **338**: 334–337, 1989.

Horn, D. and Usher, M. "Excitatory-inhibitory networks with dynamical thresholds." *Int. J. of Neural Systems* **1**: 249–257, 1990.

Koch, C. and Schuster, H. "A simple network showing burst synchronization without frequency locking." *Neural Computation*, in press.

McCulloch, W.S. and Pitts, W.A. "A logical calculus of the ideas immanent in neural nets." *Bull. Math. Biophys.* **5**: 115-137, 1943.

Niebur, E., Schuster, H.G., Kammen, D.M. and Koch, C. "Oscillator-phase coupling for different two-dimensional network connectivities." *Phys. Rev. A.*, in press.

Schuster, H.G. and Wagner, P. "A model for neuronal oscillations in the visual cortex: I Mean-field theory and the derivation of the phase equations." *Biol. Cybern.* **64**: 77-82, 1990.

Sompolinsky, H., Golomb, D. and Kleinfeld, D. "Global processing of visual stimuli in a neural network of coupled oscillators." *Proc. Natl. Acad. Sci. USA* **87**: 7200–7204, 1989.

von der Malsburg, C. and Schneider, W. "A neural cocktail-party processor." *Biol. Cybern.* **54**: 29–40, 1986.

Wilson, M.A. and Bower, J.M. "Cortical oscillations and temporal interactions in a computer simulation of piriform cortex." *J. Neurophysiol.*, in press.